# Kohonen Networks and Clustering: Comparative Performance in Color Clustering

**Wesley Snyder**

Department of Radiology
Bowman Gray School of
Medicine
Wake Forest University
Winston-Salem, NC 27103

**Daniel Nissman, David Van den Bout,
and Griff Bilbro**
Center for Communications and Signal Processing
North Carolina State University
Raleigh, NC 27695

## Abstract

The problem of color clustering is defined and shown to be a problem of assigning a large number (hundreds of thousands) of 3-vectors to a small number (256) of clusters. Finding those clusters in such a way that they best represent a full color image using only 256 distinct colors is a burdensome computational problem. In this paper, the problem is solved using "classical" techniques -- $k$-means clustering, vector quantization (which turns out to be the same thing in this application), competitive learning, and Kohonen self-organizing feature maps. Quality of the result is judged subjectively by how much the pseudo-color result resembles the true color image, by RMS quantization error, and by run time. The Kohonen map provides the best solution.

## 1 INTRODUCTION

"Clustering", "vector quantization", and "unsupervised learning" are all words which describe the same process: assigning a few exemplars to represent a large set of samples. Performing that process is the subject of a substantial body of literature. In this paper, we are concerned with the comparison of various clustering techniques to a particular, practical application: color clustering.

The color clustering problem is as follows: an image is recorded in full color -- that is, three components, RED, GREEN, and BLUE, each of which has been measured to 8 bits of precision. Thus, each pixel is a 24 bit quantity. We must find a representation in which 2563 possible colors are represented by only 8 bits per pixel. That is, for a problem with 256000 variables (512 x 512) variables, assign each variable to one of only 256 classes.

The color clustering problem is currently of major economic interest since millions of display systems are sold each year which can only store 8 bits per pixel, but on which users would like to be able to display "true" color (or at least as near true color as possible).

In this study, we have approached the problem using the standard techniques from the literature (including $k$-means -- ISODATA clustering[1,3,6], LBG[4]), competitive learning (referred to as CL herein)[2], and Kohonen feature maps[5,7,9]. The Kohonen feature map

(referred to as KFM herein) was found to win "hands down", providing both the best quality image (subjectively) and objectively (based on quantization error), as well as the fastest run times.

# 2 BACKGROUND - METHODS TESTED

In almost all clustering algorithms, we begin with some (usually ad-hoc) determination of initial cluster centers. The number of such centers generally remains the same, although some algorithms (e.g. ISODATA[10]) allow the number to evolve through the running of the algorithm. In this work, we know that we want to find 256 distinct clusters. The basic idea behind most of these methods is to update the cluster closest to the current data point by moving it some small increment towards that data point. After the data has been presented to the algorithm sufficiently often, the clusters should converge to the real cluster means. Typically, one has to cycle through the training set several times (sometimes a large number of times) to get an acceptable solution. Each run though the training set is termed an *epoch*.

## 2.1 *K*-MEANS

The well-known [6] *k*-means algorithm for clustering is as follows (see [10] for a tutorial explanation).

1. Begin with an arbitrary assignment of samples to clusters or begin with an arbitrary set of cluster centers and assign samples to nearest centers.

2. Compute the sample mean of each cluster.

3. Reassign each sample to the cluster with the nearest mean.

4. If the classification of all samples has not changed, stop; else go to step 2.

## 2.2 LBG VECTOR QUANTIZATION

In this method, 256 colors are picked randomly from the scene. These are referred to as the "codebook". Each pixel in the image is then assigned to the "nearest" entry in the codebook. After assignment of all pixels, the mean of each bin[1] is calculated. If the difference between the codebook entry and the mean of the corresponding bin is below threshold for all entries, the "optimal" codebook has been located. In [4], the algorithm is shown to work for a large variety of distance functions; however, for applications (such as this one) where the Euclidean metric is most appropriate, the algorithm becomes identical to *k*-means. In [8], results similar to those we found are reported in the color clustering problem.

## 2.3 KOHONEN MAPS AND COMPETITIVE LEARNING

In competitive learning algorithms, data examples are presented sequentially to the system. The cluster center most similar to the data example is determined, and that center is moved slightly toward the example.

The update rule for competitive learning can be described as follows:

$$\omega_i^{new} = \omega_i^{old} + n\,(data - \omega_i^{old}) \qquad \text{(EQ 1)}$$

where $w_i$ is the weight vector (or mean) corresponding to cluster $i$ and $h$ is the learning parameter (typically on the order of 0.01).

In the case of Kohonen maps, however, the algorithm is slightly more complicated. All clusters are connected to each other according to a *topological* map. When the closest cluster to a data point (the primary cluster) is updated, so are its immediate neighbors (the proximity clusters) in terms of the topological map. In feature space, it is possible, initially, for the neighbors of the primary cluster to not be its topological neighbors. By the nature of the update rule, the neighbors of the primary cluster in topological space will become its neighbors in feature space after some period of time. This is very desirable for applications in which a minimum distance between related clusters is desired (the Traveling Salesman Problem, for example).

Often, it is the case that a single cluster is chosen much of the time, if not all of the time, because of the order in which data is presented and the manner in which the clusters are initialized. In order to make clustering work in a practical context, one needs to include a term in the distance calculation which reduces the probability of updating an often-used cluster. Such a term is called the *conscience*[2]. Its effect is to increase the effective distance of a cluster from a data point. An alternative approach to the use of a conscience is to increment a counter for each cluster which has been passed over for updating and then subtract some multiple of this counter from the calculated distance. We call this the *loneliness* term, and used it because the implementation turned out to be more convenient, and the performance similar to that of conscience.

For KFM, the primary cluster is updated as indicated in Eqn. 1. The proximity clusters are updated in a similar fashion

$$\omega_j^{new} = \omega_j^{old} + F\,(\eta, d_{ij})\,(data - \omega_j^{old}) \qquad \text{(EQ 2)}$$

where $w_j$ is the weight vector corresponding to the proximity cluster $j$, $d_{ij}$ is the topological distance between clusters $i$ and $j$, and $F\,(\eta, d_{ij})$ is some decreasing function of the distance between $i$ and $j$ with a maximum at $\eta$.

## 3 Application to Color Clustering

Making no assumptions concerning the input image, we chose an appropriate topology for the KFM algorithm which would easily lend itself to describing a uniform distribution of colors in RGB space. Such a distribution is a rectangular solid in the 3-D color space. We chose the dimensions of this block to be 6x7x6 -- corresponding to 252 clusters rather than the 256 allowable -- under the assumption that the omission of those four clusters would not make a perceptible difference. The clusters were initialized as a small block positioned at the center of RGB space with the long axis in the green direction. This orientation was chosen because human eyes are most sensitive to green wavelengths and, hence, more resolution may be required along this axis. The exact initial orientation does not matter in the final solution, but was chosen to aid in speed of convergence.

In an attempt to significantly speed up training, each data point was assigned to one of the eight subcubes of RGB space, and then only a specified subset of clusters was searched for an appropriate candidate for updating. The clusters were subdivided, roughly, into eight subcubes as well. The effect of this is to decrease training time by approximately a factor of eight. Also, in the interest of processing time, only the six most immediate topological neighbors (those with a topological distance of one from the primary cluster) were updated. This same heuristic was applied for both CL and KFM experiments.

## 4 RESULTS

We applied all the techniques discussed, in various implementations, to actual color images, including·in particular, pictures of faces. Although also tested on larger images, all times given in this report are against a baseline case of a 128x128 image: three bands of input (red, green, blue -- 8 bits each), and one band (8 bits) of output, plus a lookup table output indicating what 24 bit color each of the 8 bit pattern represented. Given sufficient training, all the techniques produced pseudo-color images which were extremely lifelike. Comparing the images closely on a CRT, a trained observer will note variations in the color rendering, particularly in sparse colors (e.g. blue eyes in a facial scene), and will also observe color contouring. However, these details are subtle, and are not easily reproducible in a conference proceedings. Map files and corresponding images were generated for 5, 10, and 15 epochs using $h = 0.05$ and proximity $h = 0.00625$. Direct comparisons were made between Kohonen feature maps, competitive learning, and the results from $k$-means (and the LBG formulation of $k$-means). For the training runs using competitive learning, all clusters were initialized to random values within the unit sphere located in the center of RGB space. The conscience concept was used here.

In this section all timing comparisons are done on a Microvax 2000, although we have also run many of the same programs on a Decstation. The Decstation typically runs 10-15 times as fast as the Microvax. In order to compare techniques fairly, all timing is reported for the same image.

### 4.1 *K*-MEANS AND LBG EXPERIMENTS

The performance of $k$-means and LBG algorithms were strongly dependent on how long they were allowed to run. After approximately 90 minutes of execution of $k$-means, the results were as good (subjectively) as from Kohonen maps. In different experiments, $k$-means was started from the following initial configurations:

1. 256 points randomly (uniformly) distributed over RGB space

2. The 256 points on the main diagonal of color space (red=green=blue)

3. A uniform (3D) grid spread over RGB

4. Uniformly distributed over the surface of the color cube

5. Randomly distributed near the origin

Choice 2 gave the best overall performance, where "best" is determined by the time required to converge to a point where the resulting image looked "equally good" subjectively. *K*-means required 87 minutes to reach this *standard* quality, although it took 9 hours to completely converge (until no cluster center moved more than .5 units in one iteration).

## 4.2 EXPERIMENTS ON KOHONEN AND COMPETITIVE LEARNING

KFM gave an excellent rendering of color images. In particular, blue eyes were rendered extremely well in images of faces. Depending on the value of the conscience parameter, the competitive learning algorithm tended to rendered blue eyes as brown, since the dominant skin tones in facial images are shades of brown.

Speed comparisons. All of these runs were done on Microvaxen.

| Algorithm | Total time | Time/epoch |
| --- | --- | --- |
| Kohonen | 15:42 | 1:34 |
| Comp Learn | 8:38 | :52 |

Converting the image:
1:34 for Kohonen
4:16 for Competitve Learning

The subjective judgments of picture quality were made using the 10 epoch case of Kohonen maps as a reference. To quantitatively compare the performance of Kohonen maps and competitive learning, we computed the RMS color error:

$$E = \sum_i (v_i - c_i)^2 \qquad \text{(EQ 3)}$$

where $v_i$ is the actual color 3-vector at pixel $i$, and $c_i$ is the color represented by the mean of the cluster to which pixel $i$ is currently assigned. Plotting $E$ vs. epoch number for both Kohonen and competitive learning, we find the results in the figure below.

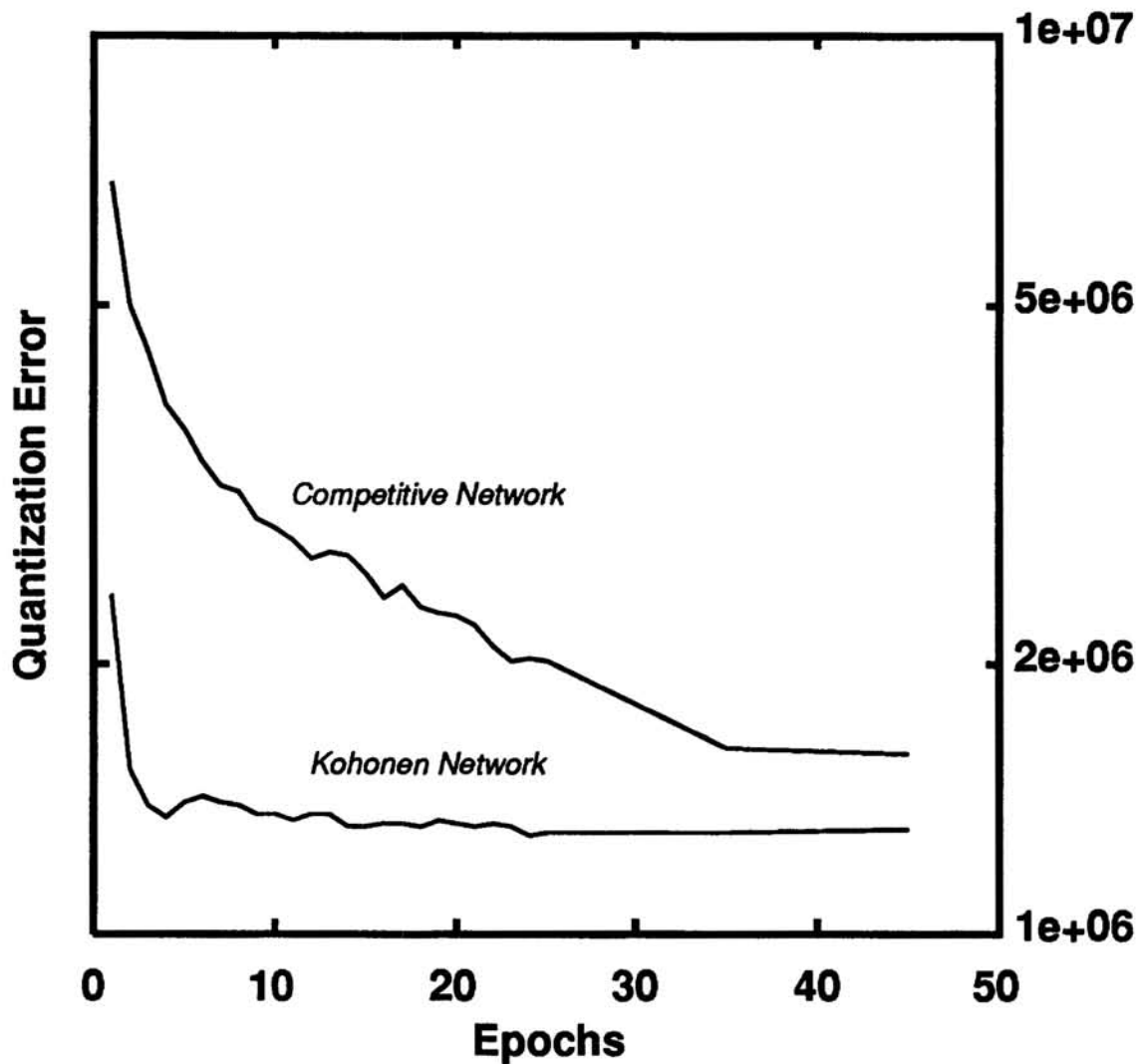

It is clear from this figure that the KFM network converges more rapidly to a stable solution with much lower error than does the competitive network. Such figures can be deceiving in image processing, however, since RMS error is a notoriously bad quality measure (small regions may have very large errors in order to make the overall average error low). In this case, however, the Kohonen map preserves the accuracy of color rendering in small regions quite well.

To evaluate the sensitivity to initial cluster center choices, both competitive learning and KFM were applied with different choices of centers. We found that competitive learning often converged to undesirable renderings, whereas KFM always yielded a good solution, even when the initial centers were all at 0,0,0.

## 5  DISCUSSION

The quality of rendering attained by these algorithms is due to the nature of facial images. There is a great deal of density in the flesh colored region and a comparatively smaller, but nonetheless siz-

able, amount in the background colors. The competitive learning algorithm found these high density regions with no problem. Greater difficulty was had with the blue eyes, since there are few examples of blue to be trained on and hence the algorithm was swamped by the high density regions. If one let the competitive learning algorithm run for a large number of epochs, it eventually found the blue cluster. The assignment of clusters to subdivisions of feature space guarantees that no region of the image was particularly emphasized, therefore allowing clusters that were solely influenced by less represented colors. However, this can also "waste" clusters in regions where there are few examples.

Furthermore, the topological structure of the Kohonen map allows one to make certain assumptions to speed up the algorithm.

Despite a minor penalty in computational speed per epoch, the Kohonen algorithm produces the image with the least error in the least amount of time. With appropriate choice of parameters, the clustered image becomes indistinguishable from the original in less than ten epochs, for essentially arbitrary initial conditions (as opposed to competitive learning). The other clustering techniques require significantly longer times.

## 6 REFERENCES

1. G. H. Ball and D. J. Hall, "ISODATA, A Novel Method of Data Analysis and Pattern Classification" SRI Technical Report (NTIS AD699616), Stanford, CA, 1965

2. D. DeSieno, "Adding a Conscience to Competitive Learning", International Conference On Neural Networks, Vol. 1, pp. 117-124, 1988

3. K. Fukunaga, *Introduction to Pattern Recognition*, Academic Press, Orlando FL, 1972

4. Y. Linde, A. Buzo, and R. Gray, "An Algorithm for Vector Quantizer Design", *IEEE Trans. Com.* Vol. COM-28, No. 1, pp. 84-95, Jan. 1980

5. T. Kohonen, "Self-Organized Formation of Topologically Correct Feature Maps", *Biological Cybernetics*, 43:56-69, 1982

6. J. Mac Queen "Some Methods for Classification and Analysis of Multivariate Observations", Proc. 5th Berkeley Symposium, 1, pp. 281-297, 1967

7. N. Nasrabadi and Y. Feng, "Vector Quantization of Images Based upon Kohonen Self-organizing Feature Maps", IEEE International Conference on Neural Networks, Vol. 1, pp. 101-108, 1986

8. H. Potlapalli, M. Jaisimha, H. Barad, A. Martinez, M. Lohrenz, J. Ryan, and J. Pollard. "Classification Techniques for Digital Map Compression" 21st Southeastern Symposium on System Theory, pp. 268-272. 1989. Tallahasee, Fl, March, 1989

9. H. Ritter and K. Schulten, "Kohonen Self-organizing Maps: Exploring their Computational Capabilities" IEEE International Conference on Neural Networks, Vol. 1, pp. 109-116, 1988

10. C. W. Therrien, *Design, Estimation, and Classification*, Wiley, NY, 1989

## Footnotes

[1] That is, all the pixels assigned to that entry in the codebook.
